# From Margin To Sparsity

**Thore Graepel, Ralf Herbrich**
Computer Science Department
Technical University of Berlin
Berlin, Germany
*{guru,ralfh}@cs.tu-berlin.de*

**Robert C. Williamson**
Department of Engineering
Australian National University
Canberra, Australia
*Bob.Williamson@anu.edu.au*

## Abstract

We present an improvement of Novikoff's perceptron convergence theorem. Reinterpreting this mistake bound as a margin dependent sparsity guarantee allows us to give a PAC–style generalisation error bound for the classifier learned by the perceptron learning algorithm. The bound value crucially depends on the margin a support vector machine would achieve on the same data set using the same kernel. Ironically, the bound yields better guarantees than are currently available for the support vector solution itself.

## 1 Introduction

In the last few years there has been a large controversy about the significance of the attained margin, i.e. the smallest real valued output of a classifiers before thresholding, as an indicator of generalisation performance. Results in the VC, PAC and luckiness frameworks seem to indicate that a large margin is a pre–requisite for small generalisation error bounds (see [14, 12]). These results caused many researchers to focus on large margin methods such as the well known support vector machine (SVM). On the other hand, the notion of *sparsity* is deemed important for generalisation as can be seen from the popularity of *Occam's razor* like arguments as well as compression considerations (see [8]).

In this paper we reconcile the two notions by reinterpreting an improved version of Novikoff's well known perceptron convergence theorem as a *sparsity guarantee* in dual space: *the existence of large margin classifiers implies the existence of sparse consistent classifiers in dual space.* Even better, this solution is easily found by the perceptron algorithm. By combining the perceptron mistake bound with a compression bound that originated from the work of Littlestone and Warmuth [8] we are able to provide a PAC like generalisation error bound for the classifier found by the perceptron algorithm whose size is determined by the magnitude of the maximally achievable margin on the dataset.

The paper is structured as follows: after introducing the perceptron in dual variables in Section 2 we improve on Novikoff's perceptron convergence bound in Section 3. Our main result is presented in the subsequent section and its consequences on the theoretical foundation of SVMs are discussed in Section 5.

## 2 (Dual) Kernel Perceptrons

We consider learning given $m$ objects $X = \{x_1, \ldots, x_m\} \in \mathcal{X}^m$ and a set $Y = \{y_1, \ldots y_m\} \in \mathcal{Y}^m$ drawn iid from a fixed distribution $\mathbf{P}_{XY} = \mathbf{P}_Z$ over the space $\mathcal{X} \times \{-1, +1\} = \mathcal{Z}$ of input–output pairs. Our hypotheses are linear classifiers $x \mapsto \text{sign}(\langle \mathbf{w}, \phi(x) \rangle)$ in some fixed *feature space* $\mathcal{K} \subseteq \ell_2^n$ where we assume that a mapping $\phi : \mathcal{X} \to \mathcal{K}$ is chosen a priori[1]. Given the features $\phi_i : \mathcal{X} \to \mathbb{R}$ the classical (primal) perceptron algorithm aims at finding a weight vector $\mathbf{w} \in \mathcal{K}$ consistent with the training data. Recently, Vapnik [14] and others — in their work on SVMs — have rediscovered that it may be advantageous to learn in the dual representation (see [1]), i.e. expanding the weight vector in terms of the training data

$$\mathbf{w}_\alpha = \sum_{i=1}^m \alpha_i \phi(x_i) = \sum_{i=1}^m \alpha_i \mathbf{x}_i \,, \tag{1}$$

and learn the $m$ expansion coefficients $\boldsymbol{\alpha} \in \mathbb{R}^m$ rather than the components of $\mathbf{w} \in \mathcal{K}$. This is particularly useful if the dimensionality $n = \dim(\mathcal{K})$ of the feature space $\mathcal{K}$ is much greater (or possibly infinite) than the number $m$ of training points.

This dual representation can be used for a rather wide class of learning algorithms (see [15]) — in particular if all we need for learning is the real valued output $\langle \mathbf{w}, \mathbf{x}_i \rangle_\mathcal{K}$ of the classifier at the $m$ training points $x_1, \ldots, x_m$. Thus it suffices to choose a symmetric function $k : \mathcal{X} \times \mathcal{X} \to \mathbb{R}$ called *kernel* and to ensure that there exists a mapping $\phi_k : \mathcal{X} \to \mathcal{K}$ such that

$$\forall x, x' \in \mathcal{X} : \quad k(x, x') = \langle \phi_k(x), \phi_k(x') \rangle_\mathcal{K} \,. \tag{2}$$

A sufficient condition is given by Mercer's theorem.

**Theorem 1 (Mercer Kernel [9, 7]).** *Any symmetric function* $k \in L_\infty(\mathcal{X} \times \mathcal{X})$ *that is positive semidefinite, i.e.*

$$\forall f \in L_2(\mathcal{X}) : \quad \int_\mathcal{X} \int_\mathcal{X} k(x, x') f(x) f(x') \, dx \, dx' \geq 0 \,,$$

*is called a Mercer kernel and has the following property: if* $\psi_i \in L_2(\mathcal{X})$ *solve the eigenvalue problem* $\int_\mathcal{X} k(x, x') \psi_i(x') \, dx' = \lambda_i \psi_i(x)$ *with* $\int_\mathcal{X} \psi_i^2(x) \, dx = 1$ *and* $\forall i \neq j : \int_\mathcal{X} \psi_i(x) \psi_j(x) \, dx = 0$ *then* $k$ *can be expanded in a uniformly convergent series, i.e.*

$$k(x, x') = \sum_{i=1}^\infty \lambda_i \psi_i(x) \psi_i(x') \,.$$

In order to see that a Mercer kernel fulfils equation (2) consider the mapping

$$\phi_k(x) = \left( \sqrt{\lambda_1} \psi_1(x), \sqrt{\lambda_2} \psi_2(x), \ldots \right) \tag{3}$$

whose existence is ensured by the third property. Finally, the perceptron learning algorithm we are going to consider is described in the following definition.

**Definition 1 (Perceptron Learning).** The *perceptron learning* procedure with the fixed learning rate $\eta \in \mathbb{R}^+$ is as follows:

1. Start in step zero, i.e. $t = 0$, with the vector $\boldsymbol{\alpha}_t = \mathbf{0}$.

2. If there exists an index $i \in \{1, \ldots, m\}$ such that $y_i \langle \mathbf{w}_{\alpha_t}, \mathbf{x}_i \rangle_\mathcal{K} \leq 0$ then

$$(\boldsymbol{\alpha}_{t+1})_i = (\boldsymbol{\alpha}_t)_i + \eta y_i \quad \Leftrightarrow \quad \mathbf{w}_{\alpha_{t+1}} = \mathbf{w}_{\alpha_t} + \eta y_i \mathbf{x}_i \,. \tag{4}$$

and $t \leftarrow t + 1$.

3. Stop, if there is no $i \in \{1, \dots, m\}$ such that $y_i \langle \mathbf{w}_{\alpha_i}, \mathbf{x}_i \rangle_{\mathcal{K}} \leq 0$.

Other variants of this algorithm have been presented elsewhere (see [2, 3]).

## 3  An Improvement of Novikoff's Theorem

In the early 60's Novikoff [10] was able to give an upper bound on the number of mistakes made by the classical perceptron learning procedure. Two years later, this bound was generalised to feature spaces using Mercer kernels by Aizerman et al. [1]. The quantity determining the upper bound is the maximally achievable unnormalised margin $\max_{\alpha \in \mathbb{R}^m} \gamma_Z(\alpha)$ normalised by the total extent $R(X)$ of the data in feature space, i.e. $R(X) = \max_{x_i \in X} \|\mathbf{x}_i\|_{\mathcal{K}}$.

**Definition 2 (Unnormalised Margin).** Given a training set $Z = (X, Y)$ and a vector $\alpha \in \mathbb{R}^m$ the *unnormalised margin* $\gamma_Z(\alpha)$ is given by

$$\gamma_Z(\alpha) = \min_{(x_i, y_i) \in Z} \frac{y_i \langle \mathbf{w}_\alpha, \mathbf{x}_i \rangle_{\mathcal{K}}}{\|\mathbf{w}_\alpha\|_{\mathcal{K}}}.$$

**Theorem 2 (Novikoffs Perceptron Convergence Theorem [10, 1]).** *Let $Z = (X, Y)$ be a training set of size $m$. Suppose that there exists a vector $\alpha^* \in \mathbb{R}^m$ such that $\gamma_Z(\alpha^*) > 0$. Then the number of mistakes made by the perceptron algorithm in Definition 1 on $Z$ is at most*

$$\left( \frac{R(X)}{\gamma_Z(\alpha^*)} \right)^2.$$

Surprisingly, this bound is highly influenced by the data point $x_i \in X$ with the largest norm $\|\mathbf{x}_i\|$ albeit rescaling of a data point would not change its classification. Let us consider rescaling of the training set $X$ before applying the perceptron algorithm. Then for the normalised training set we would have $R(X_{\text{norm}}) = 1$ and $\gamma_Z(\alpha)$ would change into the *normalised margin* $\Gamma_Z(\alpha)$ first advocated in [6].

**Definition 3 (Normalised Margin).** Given a training set $Z = (X, Y)$ and a vector $\alpha \in \mathbb{R}^m$ the *normalised margin* $\Gamma_Z(\alpha)$ is given by

$$\Gamma_Z(\alpha) = \min_{(x_i, y_i) \in Z} \frac{y_i \langle \mathbf{w}_\alpha, \mathbf{x}_i \rangle_{\mathcal{K}}}{\|\mathbf{w}_\alpha\|_{\mathcal{K}} \|\mathbf{x}_i\|_{\mathcal{K}}}.$$

By definition, for all $x_i \in X$ we have $R(X) \geq \|\mathbf{x}_i\|_{\mathcal{K}}$. Hence for any $\alpha \in \mathbb{R}^m$ and all $(x_i, y_i) \in Z$ such that $y_i \langle \mathbf{w}_\alpha, \mathbf{x}_i \rangle_{\mathcal{K}} > 0$

$$\frac{R(X)}{\frac{y_i \langle \mathbf{w}_\alpha, \mathbf{x}_i \rangle_{\mathcal{K}}}{\|\mathbf{w}_\alpha\|_{\mathcal{K}}}} \geq \frac{\|\mathbf{x}_i\|_{\mathcal{K}}}{\frac{y_i \langle \mathbf{w}_\alpha, \mathbf{x}_i \rangle_{\mathcal{K}}}{\|\mathbf{w}_\alpha\|_{\mathcal{K}}}} = \frac{1}{\frac{y_i \langle \mathbf{w}_\alpha, \mathbf{x}_i \rangle_{\mathcal{K}}}{\|\mathbf{w}_\alpha\|_{\mathcal{K}} \|\mathbf{x}_i\|_{\mathcal{K}}}},$$

which immediately implies for all $Z = (X, Y) \in \mathcal{Z}^m$ such that $\gamma_Z(\alpha) > 0$

$$\frac{R(X)}{\gamma_Z(\alpha)} \geq \frac{1}{\Gamma_Z(\alpha)}. \tag{5}$$

Thus when normalising the data in feature space, i.e.

$$k_{\text{norm}}(x, x') = \frac{k(x, x')}{\sqrt{k(x, x) \cdot k(x', x')}},$$

the upper bound on the number of steps until convergence of the classical perceptron learning procedure of Rosenblatt [11] is provably decreasing and is given by the squared r.h.s of (5).

Considering the form of the update rule (4) we observe that this result not only bounds the number of mistakes made during learning but also the number $\|\boldsymbol{\alpha}\|_0$ of non–zero coefficients in the $\boldsymbol{\alpha}$ vector. To be precise, for $\eta = 1$ it bounds the $\ell_1$ norm $\|\boldsymbol{\alpha}\|_1$ of the coefficient vector $\boldsymbol{\alpha}$ which, in turn, bounds the zero norm $\|\boldsymbol{\alpha}\|_0$ from above for all vectors with integer components. Theorem 2 thus establishes a relation between the *existence of a large margin classifier* $\mathbf{w}^*$ *and the sparseness of any solution found by the perceptron algorithm.*

## 4 Main Result

In order to exploit the guaranteed sparseness of the solution of a kernel perceptron we make use of the following lemma to be found in [8, 4].

**Lemma 1 (Compression Lemma).** *Fix $d \in \{1, \ldots, m\}$. For any measure $\mathbf{P}_Z$, the probability that $m$ examples $Z$ drawn iid according to $\mathbf{P}_Z$ will yield a classifier $\boldsymbol{\alpha}(Z)$ learned by the perceptron algorithm with $\|\boldsymbol{\alpha}(Z)\|_0 = d$ whose generalisation error $\mathbf{P}_{XY}\left[Y \left\langle \mathbf{w}_{\boldsymbol{\alpha}(Z)}, \boldsymbol{\phi}(X) \right\rangle_{\mathcal{K}} \leq 0\right]$ is greater than $\varepsilon$ is at most*

$$\binom{m}{d} (1 - \varepsilon)^{m-d} . \tag{6}$$

*Proof.* Since we restrict the solution $\boldsymbol{\alpha}(Z)$ with generalisation error greater than $\varepsilon$ only to use $d$ points $Z_d \subseteq Z$ but still to be consistent with the remaining set $Z \setminus Z_d$, this probability is at most $(1 - \varepsilon)^{m-d}$ for a fixed subset $Z_d$. The result follows by the union bound over all $\binom{m}{d}$ subsets $Z_d$. Intuitively, the consistency on the $m - d$ unused training points witnesses the small generalisation error with high probability. $\square$

If we set (6) to $\frac{\delta}{m}$ and solve for $\varepsilon$ we have that with probability at most $\frac{\delta}{m}$ over the random draw of the training set $Z$ the perceptron learning algorithm finds a vector $\boldsymbol{\alpha}$ such that $\|\boldsymbol{\alpha}\|_0 = d$ and whose generalisation error is greater than $\varepsilon(m, d) = \frac{1}{m-d}\left(\ln\left(\binom{m}{d}\right) + \ln(m) + \ln\left(\frac{1}{\delta}\right)\right)$. Thus by the union bound, if the perceptron algorithm converges, the probability that the generalisation error of its solution is greater than $\varepsilon(m, \|\boldsymbol{\alpha}\|_0)$ is at most $\delta$. We have shown the following sparsity bounds also to be found in [4].

**Theorem 3 (Generalisation Error Bound for Perceptrons).** *For any measure $\mathbf{P}_Z$, with probability at least $1 - \delta$ over the random draw of the training set $Z$ of size $m$, if the perceptron learning algorithm converges to the vector $\boldsymbol{\alpha}$ of coefficients then its generalisation error $\mathbf{P}_{XY}\left[Y \left\langle \mathbf{w}_{\boldsymbol{\alpha}(Z)}, \boldsymbol{\phi}(X) \right\rangle_{\mathcal{K}} \leq 0\right]$ is less than*

$$\frac{1}{m - \|\boldsymbol{\alpha}\|_0}\left(\ln\left(\binom{m}{\|\boldsymbol{\alpha}\|_0}\right) + \ln(m) + \ln\left(\frac{1}{\delta}\right)\right) . \tag{7}$$

This theorem in itself constitutes a powerful result and can easily be adopted to hold for a large class of learning algorithms including SVMs [4]. This bound often outperforms margin bounds for practically relevant training set sizes, e.g. $m < 100\,000$. Combining Theorem 2 and Theorem 3 thus gives our main result.

> **Theorem 4 (Margin Bound).** *For any measure* $\mathbf{P}_Z$, *with probability at least* $1 - \delta$ *over the random draw of the training set $Z$ of size $m$, if there exists a vector $\boldsymbol{\alpha}^*$ such that*
>
> $$\kappa^* = \left\lceil \left( \frac{R(X)}{\gamma_Z(\boldsymbol{\alpha}^*)} \right)^2 \right\rceil \le m$$
>
> *then the generalisation error* $\mathbf{P}_{XY}\left[ Y \langle \mathbf{w}_{\boldsymbol{\alpha}(Z)}, \phi(X) \rangle_K \le 0 \right]$ *of the classifier $\boldsymbol{\alpha}$ found by the perceptron algorithm is less than*
>
> $$\frac{1}{m - \kappa^*} \left( \ln \left( \binom{m}{\kappa^*} \right) + \ln(m) + \ln\left(\frac{1}{\delta}\right) \right). \tag{8}$$

The most intriguing feature of this result is that the *mere existence* of a large margin classifier $\boldsymbol{\alpha}^*$ is sufficient to guarantee a small generalisation error for the solution $\boldsymbol{\alpha}$ of the perceptron although its attained margin $\gamma_Z(\boldsymbol{\alpha})$ is likely to be much smaller than $\gamma_Z(\boldsymbol{\alpha}^*)$. It has long been argued that the attained margin $\gamma_Z(\boldsymbol{\alpha})$ *itself* is the crucial quantity controlling the generalisation error of $\boldsymbol{\alpha}$. In light of our new result *if there exists a consistent classifier $\boldsymbol{\alpha}^*$ with large margin we know that there also exists at least one classifier $\boldsymbol{\alpha}$ with high sparsity* that can efficiently be found using the perceptron algorithm. In fact, whenever the SVM appears to be theoretically justified by a large observed margin, every solution found by the perceptron algorithm has a small guaranteed generalisation error — mostly even smaller than current bounds on the generalisation error of SVMs. Note that for a given training sample $Z$ it is not unlikely that by permutation of $Z$ there exist $\mathcal{O}\left( \binom{m}{\kappa^*} \right)$ many different consistent sparse classifiers $\boldsymbol{\alpha}$.

## 5    Impact on the Foundations of Support Vector Machines

Support vector machines owe their popularity mainly to their theoretical justification in the learning theory. In particular, two arguments have been put forward to single out the solutions found by SVMs [14, p. 139]:

> SVM (optimal hyperplanes) can generalise because
> 1. the expectation of the data compression is large.
> 2. the expectation of the margin is large.

The second reason is often justified by margin results (see [14, 12]) which bound the generalisation of a classifier $\boldsymbol{\alpha}$ in terms of its *own attained* margin $\gamma_Z(\boldsymbol{\alpha})$. If we require the slightly stronger condition that $\kappa^* < \frac{m}{n}$, $n \ge 4$, then our bound (8) for solutions of perceptron learning can be upper bounded by

$$\frac{1}{m} \left( \kappa^* \ln \left( \frac{2em}{\kappa^*} \right) + \ln\left( m^{\frac{n}{n-1}} \right) + \ln\left( \frac{1}{\delta^{\frac{n}{n-1}}} \right) \right),$$

which has to be compared with the PAC margin bound (see [12, 5])

$$\frac{2}{m} \left( 64\kappa^* \log_2 \left( \frac{2em}{16\kappa^*} \right) \log_2(32m) + \log_2(2m) + \log_2\left( \frac{1}{\delta} \right) \right).$$

Despite the fact that the former result also holds true for the margin $\Gamma_Z(\boldsymbol{\alpha}^*)$ (which could loosely be upper bounded by (5))

- the PAC margin bound's decay (as a function of $m$) is slower by a $\log_2(32m)$ factor,

| digit | 0 | 1 | 2 | 3 | 4 | 5 | 6 | 7 | 8 | 9 |
|---|---|---|---|---|---|---|---|---|---|---|
| perceptron | 0.2 | 0.2 | 0.4 | 0.4 | 0.4 | 0.4 | 0.4 | 0.5 | 0.6 | 0.7 |
| $\|\alpha\|_0$ | 740 | 643 | 1168 | 1512 | 1078 | 1277 | 823 | 1103 | 1856 | 1920 |
| mistakes | 844 | 843 | 1345 | 1811 | 1222 | 1497 | 960 | 1323 | 2326 | 2367 |
| bound | 6.7 | 6.0 | 9.8 | 12.0 | 9.2 | 10.5 | 7.4 | 9.4 | 14.3 | 14.6 |
| SVM | 0.2 | 0.1 | 0.4 | 0.4 | 0.4 | 0.5 | 0.3 | 0.4 | 0.5 | 0.6 |
| $\|\alpha\|_0$ | 1379 | 989 | 1958 | 1900 | 1224 | 2024 | 1527 | 2064 | 2332 | 2765 |
| bound | 11.2 | 8.6 | 14.9 | 14.5 | 10.2 | 15.3 | 12.2 | 15.5 | 17.1 | 19.6 |

Table 1: Results of kernel perceptrons and SVMs on NIST (taken from [2, Table 3]). The kernel used was $k\left(x, x'\right) = \left(\langle x, x'\rangle_{\mathcal{X}} + 1\right)^4$ and $m = 60\,000$. For both algorithms we give the measured generalisation error (in %), the attained sparsity and the bound value (in %, $\delta = 0.05$) of (7).

- for any $m$ and almost any $\delta$ the margin bound given in Theorem 4 guarantees a smaller generalisation error.
- For example, using the empirical value $\kappa^* \approx 600$ (see [14, p. 153]) in the NIST handwritten digit recognition task and inserting this value into the PAC margin bound, it would need the astronomically large number of $m > 410\,743\,386$ to obtain a bound value of 0.112 as obtained by (3) for the digit "0" (see Table 1).

With regard to the first reason, it has been confirmed experimentally that SVMs find solutions which are sparse in the expansion coefficients $\alpha$. However, there cannot exist any distribution–free guarantee that the number of support vectors will in fact be small[2]. In contrast, Theorem 2 gives an explicit bound on the sparsity in terms of the achievable margin $\gamma_Z\left(\alpha^*\right)$. Furthermore, experimental results on the NIST datasets show that the sparsity of solution found by the perceptron algorithm is consistently (and often by a factor of two) greater than that of the SVM solution (see [2, Table 3] and Table 1).

## 6  Conclusion

We have shown that the generalisation error of a very simple and efficient learning algorithm for linear classifiers — the perceptron algorithm — can be bounded by a quantity involving the margin of the classifier the SVM would have found on the same training data using the same kernel. This result implies that the SVM solution is not at all singled out as being superior in terms of provable generalisation error. Also, the result indicates that *sparsity* of the solution may be a more fundamental property than the size of the attained margin (since a large value of the latter implies a large value of the former).

Our analysis raises an interesting question: having chosen a good kernel, corresponding to a metric in which inter–class distances are great and intra–class distances are short, in how far does it matter which consistent classifier we use? Experimental

results seem to indicate that a vast variety of heuristics for finding consistent classifiers, e.g. kernel Fisher discriminant, linear programming machines, Bayes point machines, kernel PCA & linear SVM, sparse greedy matrix approximation perform comparably (see http://www.kernel-machines.org/).

## Acknowledgements

This work was done while TG and RH were visiting the ANU Canberra. They would like to thank Peter Bartlett and Jon Baxter for many interesting discussions. Furthermore, we would like to thank the anonymous reviewer, Olivier Bousquet and Matthias Seeger for very useful remarks on the paper.

## Footnotes

[1]Somtimes, we abbreviate $\phi(x)$ by $\mathbf{x}$ always assuming $\phi$ is fixed.

[2]Consider a distribution $\mathbf{P}_{XY}$ on two parallel lines with support in the unit ball. Suppose that their mutual distance is $\sqrt{2}$. Then the number of support vectors equals the training set size whereas the perceptron algorithm never uses more than two points by Theorem 2. One could argue that it is the number of *essential support vectors* [13] that characterises the data compression of an SVM (which would also have been two in our example). Their determination, however, involves a combinatorial optimisation problem and can thus never be performed in practical applications.

## References

[1] M. Aizerman, E. Braverman, and L. Rozonoer. Theoretical foundations of the potential function method in pattern recognition learning. *Automation and Remote Control*, 25:821–837, 1964.

[2] Y. Freund and R. E. Schapire. Large margin classification using the perceptron algorithm. *Machine Learning*, 1999.

[3] T. Friess, N. Cristianini, and C. Campbell. The Kernel-Adatron: A fast and simple learning procedure for Support Vector Machines. In *Proceedings of the 15-th International Conference in Machine Learning*, pages 188–196, 1998.

[4] T. Graepel, R. Herbrich, and J. Shawe-Taylor. Generalisation error bounds for sparse linear classifiers. In *Proceedings of the Thirteenth Annual Conference on Computational Learning Theory*, pages 298–303, 2000. in press.

[5] R. Herbrich. *Learning Linear Classifiers - Theory and Algorithms*. PhD thesis, Technische Universität Berlin, 2000. accepted for publication by MIT Press.

[6] R. Herbrich and T. Graepel. A PAC-Bayesian margin bound for linear classifiers: Why SVMs work. In *Advances in Neural Information System Processing 13*, 2001.

[7] H. König. *Eigenvalue Distribution of Compact Operators*. Birkhäuser, Basel, 1986.

[8] N. Littlestone and M. Warmuth. Relating data compression and learnability. Technical report, University of California Santa Cruz, 1986.

[9] T. Mercer. Functions of positive and negative type and their connection with the theory of integral equations. *Transaction of London Philosophy Society (A)*, 209:415–446, 1909.

[10] A. Novikoff. On convergence proofs for perceptrons. In *Report at the Symposium on Mathematical Theory of Automata*, pages 24–26, Politechnical Institute Brooklyn, 1962.

[11] M. Rosenblatt. *Principles of neurodynamics: Perceptron and Theory of Brain Mechanisms*. Spartan-Books, Washington D.C., 1962.

[12] J. Shawe-Taylor, P. L. Bartlett, R. C. Williamson, and M. Anthony. Structural risk minimization over data–dependent hierarchies. *IEEE Transactions on Information Theory*, 44(5):1926–1940, 1998.

[13] V. Vapnik. *Statistical Learning Theory*. John Wiley and Sons, New York, 1998.

[14] V. Vapnik. *The Nature of Statistical Learning Theory*. Springer, second edition, 1999.

[15] G. Wahba. Support Vector Machines, Reproducing Kernel Hilbert Spaces and the randomized GACV. Technical report, Department of Statistics, University of Wisconsin, Madison, 1997. TR-NO-984.
